# Learning in Hilbert vs. Banach Spaces: A Measure Embedding Viewpoint

**Bharath K. Sriperumbudur**
Gatsby Unit
University College London
bharath@gatsby.ucl.ac.uk

**Kenji Fukumizu**
The Institute of Statistical
Mathematics, Tokyo
fukumizu@ism.ac.jp

**Gert R. G. Lanckriet**
Dept. of ECE
UC San Diego
gert@ece.ucsd.edu

## Abstract

The goal of this paper is to investigate the advantages and disadvantages of learning in Banach spaces over Hilbert spaces. While many works have been carried out in generalizing Hilbert methods to Banach spaces, in this paper, we consider the simple problem of learning a Parzen window classifier in a reproducing kernel Banach space (RKBS)—which is closely related to the notion of embedding probability measures into an RKBS—in order to carefully understand its pros and cons over the Hilbert space classifier. We show that while this generalization yields richer distance measures on probabilities compared to its Hilbert space counterpart, it however suffers from serious computational drawback limiting its practical applicability, which therefore demonstrates the need for developing efficient learning algorithms in Banach spaces.

## 1 Introduction

Kernel methods have been popular in machine learning and pattern analysis for their superior performance on a wide spectrum of learning tasks. They are broadly established as an easy way to construct nonlinear algorithms from linear ones, by embedding data points into reproducing kernel Hilbert spaces (RKHSs) [1, 14, 15]. Over the last few years, generalization of these techniques to Banach spaces has gained interest. This is because any two Hilbert spaces over a common scalar field with the same dimension are isometrically isomorphic while Banach spaces provide more variety in geometric structures and norms that are potentially useful for learning and approximation.

To sample the literature, classification in Banach spaces, more generally in metric spaces were studied in [3, 22, 11, 5]. Minimizing a loss function subject to a regularization condition on a norm in a Banach space was studied by [3, 13, 24, 21] and online learning in Banach spaces was considered in [17]. While all these works have focused on theoretical generalizations of Hilbert space methods to Banach spaces, the practical viability and inherent computational issues associated with the Banach space methods has so far not been highlighted. The goal of this paper is to study the advantages/disadvantages of learning in Banach spaces in comparison to Hilbert space methods, in particular, from the point of view of embedding probability measures into these spaces.

The concept of embedding probability measures into RKHS [4, 6, 9, 16] provides a powerful and straightforward method to deal with high-order statistics of random variables. An immediate application of this notion is to problems of comparing distributions based on finite samples: examples include tests of homogeneity [9], independence [10], and conditional independence [7]. Formally, suppose we are given the set $\mathscr{P}(\mathcal{X})$ of all Borel probability measures defined on the topological space $\mathcal{X}$, and the RKHS $(\mathcal{H}, k)$ of functions on $\mathcal{X}$ with $k$ as its reproducing kernel (r.k.). If $k$ is measurable and bounded, then we can embed $\mathbb{P}$ in $\mathcal{H}$ as

$$\mathbb{P} \mapsto \int_{\mathcal{X}} k(\cdot, x) \, d\mathbb{P}(x). \tag{1}$$

Given the embedding in (1), the RKHS distance between the embeddings of $\mathbb{P}$ and $\mathbb{Q}$ defines a pseudo-metric between $\mathbb{P}$ and $\mathbb{Q}$ as

$$\gamma_k(\mathbb{P}, \mathbb{Q}) := \left\| \int_{\mathcal{X}} k(\cdot, x) \, d\mathbb{P}(x) - \int_{\mathcal{X}} k(\cdot, x) \, d\mathbb{Q}(x) \right\|_{\mathcal{H}}. \tag{2}$$

It is clear that when the embedding in (1) is injective, then $\mathbb{P}$ and $\mathbb{Q}$ can be distinguished based on their embeddings $\int_{\mathcal{X}} k(\cdot, x) \, d\mathbb{P}(x)$ and $\int_{\mathcal{X}} k(\cdot, x) \, d\mathbb{Q}(x)$. [18] related RKHS embeddings to the problem of binary classification by showing that $\gamma_k(\mathbb{P}, \mathbb{Q})$ is the negative of the optimal risk associated with the Parzen window classifier in $\mathcal{H}$. Extending this classifier to Banach space and studying the highlights/issues associated with this generalization will throw light on the same associated with more complex Banach space learning algorithms. With this motivation, in this paper, we consider the generalization of the notion of RKHS embedding of probability measures to Banach spaces—in particular reproducing kernel Banach spaces (RKBSs) [24]—and then compare the properties of the RKBS embedding to its RKHS counterpart.

To derive RKHS based learning algorithms, it is essential to appeal to the Riesz representation theorem (as an RKHS is defined by the continuity of evaluation functionals), which establishes the existence of a reproducing kernel. This theorem hinges on the fact that a notion of inner product can be defined on Hilbert spaces. In this paper, as in [24], we deal with RKBSs that are *uniformly Fréchet differentiable* and *uniformly convex* (called as s.i.p. RKBS) as many Hilbert space arguments—most importantly the Riesz representation theorem—can be carried over to such spaces through the notion of *semi-inner-product* (s.i.p.) [12], which is a more general structure than an inner product. Based on Zhang et al. [24], who recently developed RKBS counterparts of RKHS based algorithms like regularization networks, support vector machines, kernel principal component analysis, etc., we provide a review of s.i.p. RKBS in Section 3. We present our main contributions in Sections 4 and 5. In Section 4, *first*, we derive an RKBS embedding of $\mathbb{P}$ into $\mathcal{B}'$ as

$$\mathbb{P} \mapsto \int_{\mathcal{X}} K(\cdot, x) \, d\mathbb{P}(x), \tag{3}$$

where $\mathcal{B}$ is an s.i.p. RKBS with $K$ as its reproducing kernel (r.k.) and $\mathcal{B}'$ is the topological dual of $\mathcal{B}$. Note that (3) is similar to (1), but more general than (1) as $K$ in (3) need not have to be positive definite (pd), in fact, not even symmetric (see Section 3; also see Examples 2 and 3). Based on (3), we define

$$\gamma_K(\mathbb{P}, \mathbb{Q}) := \left\| \int_{\mathcal{X}} K(\cdot, x) \, d\mathbb{P}(x) - \int_{\mathcal{X}} K(\cdot, x) \, d\mathbb{Q}(x) \right\|_{\mathcal{B}'},$$

a pseudo-metric on $\mathscr{P}(\mathcal{X})$, which we show to be the negative of the optimal risk associated with the Parzen window classifier in $\mathcal{B}'$. *Second*, we characterize the injectivity of (3) in Section 4.1 wherein we show that the characterizations obtained for the injectivity of (3) are similar to those obtained for (1) and coincide with the latter when $\mathcal{B}$ is an RKHS. *Third*, in Section 4.2, we consider the empirical estimation of $\gamma_K(\mathbb{P}, \mathbb{Q})$ based on finite random samples drawn i.i.d. from $\mathbb{P}$ and $\mathbb{Q}$ and study its consistency and the rate of convergence. This is useful in applications like two-sample tests (also in binary classification as it relates to the consistency of the Parzen window classifier) where different $\mathbb{P}$ and $\mathbb{Q}$ are to be distinguished based on the finite samples drawn from them and it is important that the estimator is consistent for the test to be meaningful. We show that the consistency and the rate of convergence of the estimator depend on the *Rademacher type* of $\mathcal{B}'$. This result coincides with the one obtained for $\gamma_k$ when $\mathcal{B}$ is an RKHS.

The above mentioned results, while similar to results obtained for RKHS embeddings, are significantly more general, as they apply RKBS spaces, which subsume RKHSs. We can therefore expect to obtain "richer" metrics $\gamma_K$ than when being restricted to RKHSs (see Examples 1–3). On the other hand, one disadvantage of the RKBS framework is that $\gamma_K(\mathbb{P}, \mathbb{Q})$ cannot be computed in a closed form unlike $\gamma_k$ (see Section 4.3). Though this could seriously limit the practical impact of the RKBS embeddings, in Section 5, we show that closed form expression for $\gamma_K$ and its empirical estimator can be obtained for some non-trivial Banach spaces (see Examples 1–3). However, the critical drawback of the RKBS framework is that the computation of $\gamma_K$ and its empirical estimator is significantly more involved and expensive than the RKHS framework, which means a simple kernel algorithm like a Parzen window classifier, when generalized to Banach spaces suffers from a serious computational drawback, thereby limiting its practical impact. Given the advantages of learning in Banach space over Hilbert space, this work, therefore demonstrates the need for the

development of efficient algorithms in Banach spaces in order to make the problem of learning in Banach spaces worthwhile compared to its Hilbert space counterpart. The proofs of the results in Sections 4 and 5 are provided in the supplementary material.

## 2 Notation

We introduce some notation that is used throughout the paper. For a topological space $\mathcal{X}$, $C(\mathcal{X})$ (*resp.* $C_b(\mathcal{X})$) denotes the space of all continuous (*resp.* bounded continuous) functions on $\mathcal{X}$. For a locally compact Hausdorff space $\mathcal{X}$, $f \in C(\mathcal{X})$ is said to *vanish at infinity* if for every $\epsilon > 0$ the set $\{x : |f(x)| \geq \epsilon\}$ is compact. The class of all continuous $f$ on $\mathcal{X}$ which vanish at infinity is denoted as $C_0(\mathcal{X})$. For a Borel measure $\mu$ on $\mathcal{X}$, $L^p(\mathcal{X}, \mu)$ denotes the Banach space of $p$-power ($p \geq 1$) $\mu$-integrable functions. For a function $f$ defined on $\mathbb{R}^d$, $\hat{f}$ and $f^\vee$ denote the Fourier and inverse Fourier transforms of $f$. Since $\hat{f}$ and $f^\vee$ on $\mathbb{R}^d$ can be defined in $L^1$, $L^2$ or more generally in *distributional* senses, they should be treated in the appropriate sense depending on the context. In the $L^1$ sense, the Fourier and inverse Fourier transforms of $f \in L^1(\mathbb{R}^d)$ are defined as: $\hat{f}(y) = (2\pi)^{-d/2} \int_{\mathbb{R}^d} f(x) e^{-i\langle y, x \rangle} dx$ and $f^\vee(y) = (2\pi)^{-d/2} \int_{\mathbb{R}^d} f(x) e^{i\langle y, x \rangle} dx$, where $i$ denotes the imaginary unit $\sqrt{-1}$. $\phi_{\mathbb{P}} := \int_{\mathbb{R}^d} e^{i\langle \cdot, x \rangle} d\mathbb{P}(x)$ denotes the characteristic function of $\mathbb{P}$.

## 3 Preliminaries: Reproducing Kernel Banach Spaces

In this section, we briefly review the theory of RKBSs, which was recently studied by [24] in the context of learning in Banach spaces. Let $\mathcal{X}$ be a prescribed input space.

**Definition 1** (Reproducing kernel Banach space). *An RKBS $\mathcal{B}$ on $\mathcal{X}$ is a reflexive Banach space of functions on $\mathcal{X}$ such that its topological dual $\mathcal{B}'$ is isometric to a Banach space of functions on $\mathcal{X}$ and the point evaluations are continuous linear functionals on both $\mathcal{B}$ and $\mathcal{B}'$.*

Note that if $\mathcal{B}$ is a Hilbert space, then the above definition of RKBS coincides with that of an RKHS. Let $(\cdot, \cdot)_{\mathcal{B}}$ be a bilinear form on $\mathcal{B} \times \mathcal{B}'$ wherein $(f, g^*)_{\mathcal{B}} := g^*(f)$, $f \in \mathcal{B}$, $g^* \in \mathcal{B}'$. Theorem 2 in [24] shows that if $\mathcal{B}$ is an RKBS on $\mathcal{X}$, then there exists a unique function $K : \mathcal{X} \times \mathcal{X} \to \mathbb{C}$ called the reproducing kernel (r.k.) of $\mathcal{B}$, such that the following hold:

($a_1$) $K(x, \cdot) \in \mathcal{B}$, $K(\cdot, x) \in \mathcal{B}'$, $x \in \mathcal{X}$,

($a_2$) $f(x) = (f, K(\cdot, x))_{\mathcal{B}}$, $f^*(x) = (K(x, \cdot), f^*)_{\mathcal{B}}$, $f \in \mathcal{B}$, $f^* \in \mathcal{B}'$, $x \in \mathcal{X}$.

Note that $K$ satisfies $K(x, y) = (K(x, \cdot), K(\cdot, y))_{\mathcal{B}}$ and therefore $K(\cdot, x)$ and $K(x, \cdot)$ are reproducing kernels for $\mathcal{B}$ and $\mathcal{B}'$ respectively. When $\mathcal{B}$ is an RKHS, $K$ is indeed the r.k. in the usual sense. Though an RKBS has exactly one r.k., different RKBSs may have the same r.k. (see Example 1) unlike an RKHS, where no two RKHSs can have the same r.k (by the Moore-Aronszajn theorem [4]). Due to the lack of inner product in $\mathcal{B}$ (unlike in an RKHS), it can be shown that the r.k. for a general RKBS can be any arbitrary function on $\mathcal{X} \times \mathcal{X}$ for a finite set $\mathcal{X}$ [24]. In order to have a substitute for inner products in the Banach space setting, [24] considered RKBS $\mathcal{B}$ that are uniformly Fréchet differentiable and uniformly convex (referred to as s.i.p. RKBS) as it allows Hilbert space arguments to be carried over to $\mathcal{B}$—most importantly, an analogue to the Riesz representation theorem holds (see Theorem 3)—through the notion of *semi-inner-product* (s.i.p.) introduced by [12]. In the following, we first present results related to general s.i.p. spaces and then consider s.i.p. RKBS.

**Definition 2** (S.i.p. space). *A Banach space $\mathcal{B}$ is said to be uniformly Fréchet differentiable if for all $f, g \in \mathcal{B}$, $\lim_{t \in \mathbb{R}, t \to 0} \frac{\|f + tg\|_{\mathcal{B}} - \|f\|_{\mathcal{B}}}{t}$ exists and the limit is approached uniformly for $f, g$ in the unit sphere of $\mathcal{B}$. $\mathcal{B}$ is said to be uniformly convex if for all $\epsilon > 0$, there exists a $\delta > 0$ such that $\|f + g\|_{\mathcal{B}} \leq 2 - \delta$ for all $f, g \in \mathcal{B}$ with $\|f\|_{\mathcal{B}} = \|g\|_{\mathcal{B}} = 1$ and $\|f - g\|_{\mathcal{B}} \geq \epsilon$. $\mathcal{B}$ is called an s.i.p. space if it is both uniformly Fréchet differentiable and uniformly convex.*

Note that uniform Fréchet differentiability and uniform convexity are properties of the norm associated with $\mathcal{B}$. [8, Theorem 3] has shown that if $\mathcal{B}$ is an s.i.p. space, then there exists a unique function $[\cdot, \cdot]_{\mathcal{B}} : \mathcal{B} \times \mathcal{B} \to \mathbb{C}$, called the semi-inner-product such that for all $f, g, h \in \mathcal{B}$ and $\lambda \in \mathbb{C}$:

($a_3$) $[f + g, h]_{\mathcal{B}} = [f, h]_{\mathcal{B}} + [g, h]_{\mathcal{B}}$,

($a_4$) $[\lambda f, g]_{\mathcal{B}} = \lambda[f, g]_{\mathcal{B}}$, $[f, \lambda g]_{\mathcal{B}} = \overline{\lambda}[f, g]_{\mathcal{B}}$,

($a_5$) $[f, f]_{\mathcal{B}} =: \|f\|_{\mathcal{B}}^2 > 0$ for $f \neq 0$,

($a_6$) (Cauchy-Schwartz) $|[f, g]_\mathcal{B}|^2 \leq \|f\|_\mathcal{B}^2 \|g\|_\mathcal{B}^2$,

and $\lim_{t \in \mathbb{R}, t \to 0} \frac{\|f + tg\|_\mathcal{B} - \|f\|_\mathcal{B}}{t} = \frac{\mathrm{Re}([g,f]_\mathcal{B})}{\|f\|_\mathcal{B}}$, $f, g \in \mathcal{B}$, $f \neq 0$, where $\mathrm{Re}(\alpha)$ and $\overline{\alpha}$ represent the real part and complex conjugate of a complex number $\alpha$. Note that s.i.p. in general do not satisfy conjugate symmetry, $[f, g]_\mathcal{B} = \overline{[g, f]_\mathcal{B}}$ for all $f, g \in \mathcal{B}$ and therefore is not linear in the second argument, unless $\mathcal{B}$ is a Hilbert space, in which case the s.i.p. coincides with the inner product.

Suppose $\mathcal{B}$ is an s.i.p. space. Then for each $h \in \mathcal{B}$, $f \mapsto [f, h]_\mathcal{B}$ defines a continuous linear functional on $\mathcal{B}$, which can be identified with a unique element $h^* \in \mathcal{B}'$, called the *dual function* of $h$. By this definition of $h^*$, we have $h^*(f) = (f, h^*)_\mathcal{B} = [f, h]_\mathcal{B}$, $f, h \in \mathcal{B}$. Using the structure of s.i.p., [8, Theorem 6] provided the following analogue in $\mathcal{B}$ to the Riesz representation theorem of Hilbert spaces.

**Theorem 3** ([8]). *Suppose $\mathcal{B}$ is an s.i.p. space. Then*

($a_7$) *(Riesz representation theorem) For each $g \in \mathcal{B}'$, there exists a unique $h \in \mathcal{B}$ such that $g = h^*$, i.e., $g(f) = [f, h]_\mathcal{B}$, $f \in \mathcal{B}$ and $\|g\|_{\mathcal{B}'} = \|h\|_\mathcal{B}$.*

($a_8$) *$\mathcal{B}'$ is an s.i.p. space with respect to the s.i.p. defined by $[h^*, f^*]_{\mathcal{B}'} := [f, h]_\mathcal{B}$, $f, h \in \mathcal{B}$ and $\|h^*\|_{\mathcal{B}'} := [h^*, h^*]_{\mathcal{B}'}^{1/2}$.*

For more details on s.i.p. spaces, we refer the reader to [8]. A concrete example of an s.i.p. space is as follows, which will prove to be useful in Section 5. Let $(\mathcal{X}, \mathscr{A}, \mu)$ be a measure space and $\mathcal{B} := L^p(\mathcal{X}, \mu)$ for some $p \in (1, +\infty)$. It is an s.i.p. space with dual $\mathcal{B}' := L^q(\mathcal{X}, \mu)$ where $q = \frac{p}{p-1}$. For each $f \in \mathcal{B}$, its dual element in $\mathcal{B}'$ is $f^* = \frac{\overline{f}|f|^{p-2}}{\|f\|_{L^p(\mathcal{X}, \mu)}^{p-2}}$. Consequently, the semi-inner-product on $\mathcal{B}$ is

$$[f, g]_\mathcal{B} = g^*(f) = \frac{\int_\mathcal{X} f \overline{g} |g|^{p-2} \, d\mu}{\|g\|_{L^p(\mathcal{X}, \mu)}^{p-2}}. \tag{4}$$

Having introduced s.i.p. spaces, we now discuss s.i.p. RKBS which was studied by [24]. Using the Riesz representation for s.i.p. spaces (see ($a_7$)), Theorem 9 in [24] shows that if $\mathcal{B}$ is an s.i.p. RKBS, then there exists a unique r.k. $K : \mathcal{X} \times \mathcal{X} \to \mathbb{C}$ and a s.i.p. kernel $G : \mathcal{X} \times \mathcal{X} \to \mathbb{C}$ such that:

($a_9$) $G(x, \cdot) \in \mathcal{B}$ for all $x \in \mathcal{X}$, $K(\cdot, x) = (G(x, \cdot))^*$, $x \in \mathcal{X}$,

($a_{10}$) $f(x) = [f, G(x, \cdot)]_\mathcal{B}$, $f^*(x) = [K(x, \cdot), f]_\mathcal{B}$ for all $f \in \mathcal{B}$, $x \in \mathcal{X}$.

It is clear that $G(x, y) = [G(x, \cdot), G(y, \cdot)]_\mathcal{B}$, $x, y \in \mathcal{X}$. Since s.i.p. in general do not satisfy conjugate symmetry, $G$ need not be Hermitian nor pd [24, Section 4.3]. The r.k. $K$ and the s.i.p. kernel $G$ coincide when $\mathrm{span}\{G(x, \cdot) : x \in \mathcal{X}\}$ is dense in $\mathcal{B}$, which is the case when $\mathcal{B}$ is an RKHS [24, Theorems 2, 10 and 11]. This means when $\mathcal{B}$ is an RKHS, then the conditions ($a_9$) and ($a_{10}$) reduce to the well-known reproducing properties of an RKHS with the s.i.p. reducing to an inner product.

## 4 RKBS Embedding of Probability Measures

In this section, we present our main contributions of deriving and analyzing the RKBS embedding of probability measures, which generalize the theory of RKHS embeddings. First, we would like to remind the reader that the RKHS embedding in (1) can be derived by choosing $\mathcal{F} = \{f : \|f\|_\mathcal{H} \leq 1\}$ in

$$\gamma_\mathcal{F}(\mathbb{P}, \mathbb{Q}) = \sup_{f \in \mathcal{F}} \left| \int_\mathcal{X} f \, d\mathbb{P} - \int_\mathcal{X} f \, d\mathbb{Q} \right|.$$

See [19, 20] for details. Similar to the RKHS case, in Theorem 4, we show that the RKBS embeddings can be obtained by choosing $\mathcal{F} = \{f : \|f\|_\mathcal{B} \leq 1\}$ in $\gamma_\mathcal{F}(\mathbb{P}, \mathbb{Q})$. Interestingly, though $\mathcal{B}$ does not have an inner product, it can be seen that the structure of semi-inner-product is sufficient enough to generate an embedding similar to (1).

**Theorem 4.** *Let $\mathcal{B}$ be an s.i.p. RKBS defined on a measurable space $\mathcal{X}$ with $G$ as the s.i.p. kernel and $K$ as the reproducing kernel with both $G$ and $K$ being measurable. Let $\mathcal{F} = \{f : \|f\|_\mathcal{B} \leq 1\}$ and $G$ be bounded. Then*

$$\gamma_K(\mathbb{P}, \mathbb{Q}) := \gamma_\mathcal{F}(\mathbb{P}, \mathbb{Q}) = \left\| \int_\mathcal{X} K(\cdot, x) \, d\mathbb{P}(x) - \int_\mathcal{X} K(\cdot, x) \, d\mathbb{Q}(x) \right\|_{\mathcal{B}'}. \tag{5}$$

Based on Theorem 4, it is clear that $\mathbb{P}$ can be seen as being embedded into $\mathcal{B}'$ as $\mathbb{P} \mapsto \int_{\mathcal{X}} K(\cdot, x)\, d\mathbb{P}(x)$ and $\gamma_K(\mathbb{P}, \mathbb{Q})$ is the distance between the embeddings of $\mathbb{P}$ and $\mathbb{Q}$. Therefore, we arrive at an embedding which looks similar to (1) and coincides with (1) when $\mathcal{B}$ is an RKHS.

Given these embeddings, two questions that need to be answered for these embeddings to be practically useful are: ($\star$) When is the embedding injective? and ($\star\star$) Can $\gamma_K(\mathbb{P}, \mathbb{Q})$ in (5) be estimated consistently and computed efficiently from finite random samples drawn i.i.d. from $\mathbb{P}$ and $\mathbb{Q}$? The significance of ($\star$) is that if (3) is injective, then such an embedding can be used to differentiate between different $\mathbb{P}$ and $\mathbb{Q}$, which can then be used in applications like two-sample tests to differentiate between $\mathbb{P}$ and $\mathbb{Q}$ based on samples drawn i.i.d. from them if the answer to ($\star\star$) is affirmative. These questions are answered in the following sections.

Before that, we show how these questions are important in binary classification. Following [18], it can be shown that $\gamma_K$ is the negative of the optimal risk associated with a Parzen window classifier in $\mathcal{B}'$, that separates the class-conditional distributions $\mathbb{P}$ and $\mathbb{Q}$ (refer to the supplementary material for details). This means that if (3) is not injective, then the maximum risk is attained for $\mathbb{P} \neq \mathbb{Q}$, i.e., distinct distributions are not classifiable. Therefore, the injectivity of (3) is of primal importance in applications. In addition, the question in ($\star\star$) is critical as well, as it relates to the consistency of the Parzen window classifier.

## 4.1 When is (3) injective?

The following result provides various characterizations for the injectivity of (3), which are similar (but more general) to those obtained for the injectivity of (1) and coincide with the latter when $\mathcal{B}$ is an RKHS.

**Theorem 5** (Injectivity of $\gamma_K$). *Suppose $\mathcal{B}$ is an s.i.p. RKBS defined on a topological space $\mathcal{X}$ with $K$ and $G$ as its r.k. and s.i.p. kernel respectively. Then the following hold:*

*(a) Let $\mathcal{X}$ be a Polish space that is also locally compact Hausdorff. Suppose $G$ is bounded and $K(x, \cdot) \in C_0(\mathcal{X})$ for all $x \in \mathcal{X}$. Then (3) is injective if $\mathcal{B}$ is dense in $C_0(\mathcal{X})$.*

*(b) Suppose the conditions in (a) hold. Then (3) is injective if $\mathcal{B}$ is dense in $L^p(\mathcal{X}, \mu)$ for any Borel probability measure $\mu$ on $\mathcal{X}$ and some $p \in [1, \infty)$.*

Since it is not easy to check for the denseness of $\mathcal{B}$ in $C_0(\mathcal{X})$ or $L^p(\mathcal{X}, \mu)$, in Theorem 6, we present an easily checkable characterization for the injectivity of (3) when $K$ is bounded continuous and translation invariant on $\mathbb{R}^d$. Note that Theorem 6 generalizes the characterization (see [19, 20]) for the injectivity of RKHS embedding (in (1)).

**Theorem 6** (Injectivity of $\gamma_K$ for translation invariant $K$). *Let $\mathcal{X} = \mathbb{R}^d$. Suppose $K(x, y) = \psi(x - y)$, where $\psi : \mathbb{R}^d \to \mathbb{R}$ is of the form $\psi(x) = \int_{\mathbb{R}^d} e^{i\langle x, \omega \rangle}\, d\Lambda(\omega)$ and $\Lambda$ is a finite complex-valued Borel measure on $\mathbb{R}^d$. Then (3) is injective if $\operatorname{supp}(\Lambda) = \mathbb{R}^d$. In addition if $K$ is symmetric, then the converse holds.*

**Remark 7.** *If $\psi$ in Theorem 6 is a real-valued pd function, then by Bochner's theorem, $\Lambda$ has to be real, nonnegative and symmetric, i.e., $\Lambda(d\omega) = \Lambda(-d\omega)$. Since $\psi$ need not be a pd function for $K$ to be a real, symmetric r.k. of $\mathcal{B}$, $\Lambda$ need not be nonnegative. More generally, if $\psi$ is a real-valued function on $\mathbb{R}^d$, then $\Lambda$ is conjugate symmetric, i.e., $\overline{\Lambda(d\omega)} = \Lambda(-d\omega)$. An example of a translation invariant, real and symmetric (but not pd) r.k. that satisfies the conditions of Theorem 6 can be obtained with $\psi(x) = (4x^6 + 9x^4 - 18x^2 + 15)\exp(-x^2)$. See Example 3 for more details.*

## 4.2 Consistency Analysis

Consider a two-sample test, wherein given two sets of random samples, $\{X_j\}_{j=1}^m$ and $\{Y_j\}_{j=1}^n$ drawn i.i.d. from distributions $\mathbb{P}$ and $\mathbb{Q}$ respectively, it is required to test whether $\mathbb{P} = \mathbb{Q}$ or not. Given a metric, $\gamma_K$ on $\mathscr{P}(\mathcal{X})$, the problem can equivalently be posed as testing for $\gamma_K(\mathbb{P}, \mathbb{Q}) = 0$ or not, based on $\{X_j\}_{j=1}^m$ and $\{Y_j\}_{j=1}^n$, in which case, $\gamma_K(\mathbb{P}, \mathbb{Q})$ is estimated based on these random samples. For the test to be meaningful, it is important that this estimate of $\gamma_K$ is consistent. [9] showed that $\gamma_K(\mathbb{P}_m, \mathbb{Q}_n)$ is a consistent estimator of $\gamma_K(\mathbb{P}, \mathbb{Q})$ when $\mathcal{B}$ is an RKHS, where $\mathbb{P}_m := \frac{1}{m}\sum_{j=1}^m \delta_{X_j}$, $\mathbb{Q}_n := \frac{1}{n}\sum_{j=1}^n \delta_{Y_j}$ and $\delta_x$ represents the Dirac measure at $x \in X$. Theorem 9 generalizes the consistency result in [9] by showing that $\gamma_K(\mathbb{P}_m, \mathbb{Q}_n)$ is a consistent estimator of

$\gamma_K(\mathbb{P}, \mathbb{Q})$ and the rate of convergence is $O(m^{(1-t)/t} + n^{(1-t)/t})$ if $\mathcal{B}'$ is of *type t*, $1 < t \leq 2$. Before we present the result, we define the *type* of a Banach space, $\mathcal{B}$ [2, p. 303].

**Definition 8** (Rademacher type of $\mathcal{B}$). *Let $1 \leq t \leq 2$. A Banach space $\mathcal{B}$ is said to be of $t$-Rademacher (or, more shortly, of type $t$) if there exists a constant $C^*$ such that for any $N \geq 1$ and any $\{f_j\}_{j=1}^N \subset \mathcal{B}$: $(\mathbb{E}\| \sum_{j=1}^N \varrho_j f_j \|_{\mathcal{B}}^t)^{1/t} \leq C^*(\sum_{j=1}^N \|f_j\|_{\mathcal{B}}^t)^{1/t}$, where $\{\varrho_j\}_{j=1}^N$ are i.i.d. Rademacher (symmetric $\pm 1$-valued) random variables.*

Clearly, every Banach space is of type 1. Since having type $t'$ for $t' > t$ implies having type $t$, let us define $t^*(\mathcal{B}) := \sup\{t : \mathcal{B} \text{ has type } t\}$.

**Theorem 9** (Consistency of $\gamma_K(\mathbb{P}_m, \mathbb{Q}_n)$). *Let $\mathcal{B}$ be an s.i.p. RKBS. Assume $\nu := \sup\{\sqrt{G(x,x)} : x \in \mathcal{X}\} < \infty$. Fix $\delta \in (0, 1)$. Then with probability $1 - \delta$ over the choice of samples $\{X_j\}_{j=1}^m \overset{i.i.d.}{\sim} \mathbb{P}$ and $\{Y_j\}_{j=1}^n \overset{i.i.d.}{\sim} \mathbb{Q}$, we have*

$$|\gamma_K(\mathbb{P}_m, \mathbb{Q}_n) - \gamma_K(\mathbb{P}, \mathbb{Q})| \leq 2C^* \nu \left( m^{\frac{1-t}{t}} + n^{\frac{1-t}{t}} \right) + \sqrt{18\nu^2 \log(4/\delta)} \left( m^{-\frac{1}{2}} + n^{-\frac{1}{2}} \right),$$

*where $t = t^*(\mathcal{B}')$ and $C^*$ is some universal constant.*

It is clear from Theorem 9 that if $t^*(\mathcal{B}') \in (1, 2]$, then $\gamma_K(\mathbb{P}_m, \mathbb{Q}_n)$ is a consistent estimator of $\gamma_K(\mathbb{P}, \mathbb{Q})$. In addition, the best rate is obtained if $t^*(\mathcal{B}') = 2$, which is the case if $\mathcal{B}$ is an RKHS. In Section 5, we will provide examples of s.i.p. RKBSs that satisfy $t^*(\mathcal{B}') = 2$.

### 4.3 Computation of $\gamma_K(\mathbb{P}, \mathbb{Q})$

We now consider the problem of computing $\gamma_K(\mathbb{P}, \mathbb{Q})$ and $\gamma_K(\mathbb{P}_m, \mathbb{Q}_n)$. Define $\lambda_{\mathbb{P}}^* := \int_{\mathcal{X}} K(\cdot, x) \, d\mathbb{P}(x)$. Consider

$$
\begin{aligned}
\gamma_K^2(\mathbb{P}, \mathbb{Q}) &= \|\lambda_{\mathbb{P}}^* - \lambda_{\mathbb{Q}}^*\|_{\mathcal{B}'}^2 \overset{(a_5)}{=} [\lambda_{\mathbb{P}}^* - \lambda_{\mathbb{Q}}^*, \lambda_{\mathbb{P}}^* - \lambda_{\mathbb{Q}}^*]_{\mathcal{B}'} \overset{(a_3)}{=} [\lambda_{\mathbb{P}}^*, \lambda_{\mathbb{P}}^* - \lambda_{\mathbb{Q}}^*]_{\mathcal{B}'} - [\lambda_{\mathbb{Q}}^*, \lambda_{\mathbb{P}}^* - \lambda_{\mathbb{Q}}^*]_{\mathcal{B}'} \\
&= \left[ \int_{\mathcal{X}} K(\cdot, x) \, d\mathbb{P}(x), \lambda_{\mathbb{P}}^* - \lambda_{\mathbb{Q}}^* \right]_{\mathcal{B}'} - \left[ \int_{\mathcal{X}} K(\cdot, x) \, d\mathbb{Q}(x), \lambda_{\mathbb{P}}^* - \lambda_{\mathbb{Q}}^* \right]_{\mathcal{B}'} \\
&\overset{(*)}{=} \int_{\mathcal{X}} [K(\cdot, x), \lambda_{\mathbb{P}}^* - \lambda_{\mathbb{Q}}^*]_{\mathcal{B}'} \, d\mathbb{P}(x) - \int_{\mathcal{X}} [K(\cdot, x), \lambda_{\mathbb{P}}^* - \lambda_{\mathbb{Q}}^*]_{\mathcal{B}'} \, d\mathbb{Q}(x) \\
&= \int_{\mathcal{X}} \left[ K(\cdot, x), \int_{\mathcal{X}} K(\cdot, y) \, d(\mathbb{P} - \mathbb{Q})(y) \right]_{\mathcal{B}'} d(\mathbb{P} - \mathbb{Q})(x), \quad (6)
\end{aligned}
$$

where $(*)$ is proved in the supplementary material. (6) is not reducible as the s.i.p. is not linear in the second argument unless $\mathcal{B}$ is a Hilbert space. This means $\gamma_K(\mathbb{P}, \mathbb{Q})$ is not representable in terms of the kernel function, $K(x, y)$ unlike in the case of $\mathcal{B}$ being an RKHS, in which case the s.i.p. in (6) reduces to an inner product providing

$$\gamma_K^2(\mathbb{P}, \mathbb{Q}) = \iint_{\mathcal{X}} K(x, y) \, d(\mathbb{P} - \mathbb{Q})(x) \, d(\mathbb{P} - \mathbb{Q})(y).$$

Since this issue holds for any $\mathbb{P}, \mathbb{Q} \in \mathscr{P}(\mathcal{X})$, it also holds for $\mathbb{P}_m$ and $\mathbb{Q}_n$, which means $\gamma_K(\mathbb{P}_m, \mathbb{Q}_n)$ cannot be computed in a closed form in terms of the kernel, $K(x, y)$ unlike in the case of an RKHS where $\gamma_K(\mathbb{P}_m, \mathbb{Q}_n)$ can be written as a simple V-statistic that depends only on $K(x, y)$ computed at $\{X_j\}_{j=1}^m$ and $\{Y_j\}_{j=1}^n$. This is one of the main drawbacks of the RKBS approach where the s.i.p. structure does not allow closed form representations in terms of the kernel $K$ (also see [24] where regularization algorithms derived in RKBS are not solvable unlike in an RKHS), and therefore could limit its practical viability. However, in the following section, we present non-trivial examples of s.i.p. RKBSs for which $\gamma_K(\mathbb{P}, \mathbb{Q})$ and $\gamma_K(\mathbb{P}_m, \mathbb{Q}_n)$ can be obtained in closed forms.

## 5 Concrete Examples of RKBS Embeddings

In this section, we present examples of RKBSs and then derive the corresponding $\gamma_K(\mathbb{P}, \mathbb{Q})$ and $\gamma_K(\mathbb{P}_m, \mathbb{Q}_n)$ in closed forms. To elaborate, we present three examples that cover the spectrum: Example 1 deals with RKBS (in fact a family of RKBSs induced by the same r.k.) whose r.k. is pd, Example 2 with RKBS whose r.k. is not symmetric and therefore not pd and Example 3 with RKBS whose r.k. is symmetric but not pd. These examples show that the Banach space embeddings result in richer metrics on $\mathscr{P}(\mathcal{X})$ than those obtained through RKHS embeddings.

**Example 1** ($K$ is positive definite). *Let $\mu$ be a finite nonnegative Borel measure on $\mathbb{R}^d$. Then for any $1 < p < \infty$ with $q = \frac{p}{p-1}$*

$$\mathcal{B}_p^{\text{pd}}(\mathbb{R}^d) := \left\{ f_u(x) = \int_{\mathbb{R}^d} u(t)e^{i\langle x,t\rangle} \, d\mu(t) \, : \, u \in L^p(\mathbb{R}^d, \mu), \, x \in \mathbb{R}^d \right\}, \qquad (7)$$

*is an RKBS with $K(x,y) = G(x,y) = (\mu(\mathbb{R}^d))^{(p-2)/p} \int_{\mathbb{R}^d} e^{-i\langle x-y,t\rangle} \, d\mu(t)$ as the r.k. and*

$$\gamma_K(\mathbb{P}, \mathbb{Q}) = \left\| \int_{\mathbb{R}^d} e^{i\langle x,\cdot\rangle} \, d(\mathbb{P} - \mathbb{Q})(x) \right\|_{L^q(\mathbb{R}^d,\mu)} = \|\phi_{\mathbb{P}} - \phi_{\mathbb{Q}}\|_{L^q(\mathbb{R}^d,\mu)}. \qquad (8)$$

*First note that $K$ is a translation invariant pd kernel on $\mathbb{R}^d$ as it is the Fourier transform of a nonnegative finite Borel measure, $\mu$, which follows from Bochner's theorem. Therefore, though the s.i.p. kernel and the r.k. of an RKBS need not be symmetric, the space in (7) is an interesting example of an RKBS, which is induced by a pd kernel. In particular, it can be seen that many RKBSs ($\mathcal{B}_p^{\text{pd}}(\mathbb{R}^d)$ for any $1 < p < \infty$) have the same r.k (ignoring the scaling factor which can be made one for any $p$ by choosing $\mu$ to be a probability measure). Second, note that $\mathcal{B}_p^{\text{pd}}$ is an RKHS when $p = q = 2$ and therefore (8) generalizes $\gamma_k(\mathbb{P}, \mathbb{Q}) = \|\phi_{\mathbb{P}} - \phi_{\mathbb{Q}}\|_{L^2(\mathbb{R}^d,\mu)}$. By Theorem 6, it is clear that $\gamma_K$ in (8) is a metric on $\mathscr{P}(\mathbb{R}^d)$ if and only if $\text{supp}(\mu) = \mathbb{R}^d$. Refer to the supplementary material for an interpretation of $\mathcal{B}_p^{\text{pd}}(\mathbb{R}^d)$ as a generalization of Sobolev space [23, Chapter 10].*

**Example 2** ($K$ is not symmetric). *Let $\mu$ be a finite nonnegative Borel measure such that its moment-generating function, i.e., $\mathcal{M}_\mu(x) := \int_{\mathbb{R}^d} e^{\langle x,t\rangle} \, d\mu(t)$ exists. Then for any $1 < p < \infty$ with $q = \frac{p}{p-1}$*

$$\mathcal{B}_p^{\text{ns}}(\mathbb{R}^d) := \left\{ f_u(x) = \int_{\mathbb{R}^d} u(t)e^{\langle x,t\rangle} \, d\mu(t) \, : \, u \in L^p(\mathbb{R}^d, \mu), \, x \in \mathbb{R}^d \right\}$$

*is an RKBS with $K(x,y) = G(x,y) = (\mathcal{M}_\mu(qx))^{(p-2)/p} \mathcal{M}_\mu(x(q-1) + y)$ as the r.k. Suppose $\mathbb{P}$ and $\mathbb{Q}$ are such that $\mathcal{M}_{\mathbb{P}}$ and $\mathcal{M}_{\mathbb{Q}}$ exist. Then $\gamma_K(\mathbb{P}, \mathbb{Q}) = \| \int_{\mathbb{R}^d} e^{\langle x,\cdot\rangle} \, d(\mathbb{P} - \mathbb{Q})(x) \|_{L^q(\mathbb{R}^d,\mu)} = \|\mathcal{M}_{\mathbb{P}} - \mathcal{M}_{\mathbb{Q}}\|_{L^q(\mathbb{R}^d,\mu)}$, which is the weighted $L^q$ distance between the moment-generating functions of $\mathbb{P}$ and $\mathbb{Q}$. It is easy to see that if $\text{supp}(\mu) = \mathbb{R}^d$, then $\gamma_K(\mathbb{P}, \mathbb{Q}) = 0 \Rightarrow \mathcal{M}_{\mathbb{P}} = \mathcal{M}_{\mathbb{Q}}$ a.e. $\Rightarrow \mathbb{P} = \mathbb{Q}$, which means $\gamma_K$ is a metric on $\mathscr{P}(\mathbb{R}^d)$. Note that $K$ is not symmetric (for $q \neq 2$) and therefore is not pd. When $p = q = 2$, $K(x,y) = \mathcal{M}_\mu(x + y)$ is pd and $\mathcal{B}_p^{\text{ns}}(\mathbb{R}^d)$ is an RKHS.*

**Example 3** ($K$ is symmetric but not positive definite). *Let $\psi(x) = Ae^{-x^2}\left(4x^6 + 9x^4 - 18x^2 + 15\right)$ with $A := (1/243)\left(4\pi^2/25\right)^{1/6}$. Then*

$$\mathcal{B}_{\frac{3}{2}}^{\text{snpd}}(\mathbb{R}) := \left\{ f_u(x) = \int_{\mathbb{R}} (x-t)^2 e^{-\frac{3(x-t)^2}{2}} u(t)\, dt \, : \, u \in L^{\frac{3}{2}}(\mathbb{R}), \, x \in \mathbb{R} \right\}$$

*is an RKBS with r.k. $K(x,y) = g(x,y) = \psi(x - y)$. Clearly, $\psi$ and therefore $K$ are not pd (though symmetric on $\mathbb{R}$) as $\widehat{\psi}(x) = \frac{-e^{-\frac{x^2}{4}}}{34992\sqrt{2}}\left(x^6 - 39x^4 + 216x^2 - 324\right)$ is not nonnegative at every $x \in \mathbb{R}$. Refer to the supplementary material for the derivation of $K$ and $\widehat{\psi}$. In addition, $\gamma_K(\mathbb{P}, \mathbb{Q}) = \| \int_{\mathbb{R}} \theta(\cdot - x)\, d(\mathbb{P} - \mathbb{Q})(x)\|_{L^q(\mathbb{R})} = \|(\widehat{\theta}\,(\overline{\phi_{\mathbb{P}}} - \overline{\phi_{\mathbb{Q}}}))^\vee\|_{L^q(\mathbb{R})}$, where $\theta(t) = t^2 e^{-\frac{3}{2}t^2}$. Since $\text{supp}(\widehat{\theta}) = \mathbb{R}$, we have $\gamma_K(\mathbb{P}, \mathbb{Q}) = 0 \Rightarrow (\widehat{\theta}\,(\overline{\phi_{\mathbb{P}}} - \overline{\phi_{\mathbb{Q}}}))^\vee = 0 \Rightarrow \widehat{\theta}\,(\phi_{\mathbb{P}} - \phi_{\mathbb{Q}}) = 0 \Rightarrow \phi_{\mathbb{P}} = \phi_{\mathbb{Q}}$ a.e., which implies $\mathbb{P} = \mathbb{Q}$ and therefore $\gamma_K$ is a metric on $\mathscr{P}(\mathbb{R})$.*

So far, we have presented different examples of RKBSs, wherein we have demonstrated the nature of the r.k., derived the Banach space embeddings in closed form and studied the conditions under which it is injective. These examples also show that the RKBS embeddings result in richer distance measures on probabilities compared to those obtained by the RKHS embeddings—an advantage gained by moving from Hilbert to Banach spaces. Now, we consider the problem of computing $\gamma_K(\mathbb{P}_m, \mathbb{Q}_n)$ in closed form and its consistency. In Section 4.3, we showed that $\gamma_K(\mathbb{P}_m, \mathbb{Q}_n)$ does not have a nice closed form expression unlike in the case of $\mathcal{B}$ being an RKHS. However, in the following, we show that for $K$ in Examples 1–3, $\gamma_K(\mathbb{P}_m, \mathbb{Q}_n)$ has a closed form expression for certain choices of $q$. Let us consider the estimation of $\gamma_K(\mathbb{P}, \mathbb{Q})$:

$$\gamma_K^q(\mathbb{P}_m, \mathbb{Q}_n) = \left\| \int_{\mathcal{X}} b(x,\cdot)\, d(\mathbb{P}_m - \mathbb{Q}_n)(x) \right\|_{L^q(\mathcal{X},\mu)}^q = \int_{\mathcal{X}} \left| \int_{\mathcal{X}} b(x,t)\, d(\mathbb{P}_m - \mathbb{Q}_n)(x) \right|^q d\mu(t)$$

$$= \int_{\mathcal{X}} \left| \frac{1}{m}\sum_{j=1}^m b(X_j, t) - \frac{1}{n}\sum_{j=1}^n b(Y_j, t) \right|^q d\mu(t), \qquad (9)$$

where $b(x, t) = e^{i\langle x,t\rangle}$ in Example 1, $b(x, t) = e^{\langle x,t\rangle}$ in Example 2 and $b(x, t) = \theta(x - t)$ with $q = 3$ and $\mu$ being the Lebesgue measure in Example 3. Since the duals of RKBSs considered in Examples 1–3 are of type $\min(q, 2)$ for $1 \le q \le \infty$ [2, p. 304], by Theorem 9, $\gamma_K(\mathbb{P}_m, \mathbb{Q}_n)$ estimates $\gamma_K(\mathbb{P}, \mathbb{Q})$ consistently at a convergence rate of $O(m^{\frac{\max(1-q,-1)}{\min(q,2)}} + n^{\frac{\max(1-q,-1)}{\min(q,2)}})$ for $q \in (1, \infty)$, with the best rate of $O(m^{-1/2} + n^{-1/2})$ attainable when $q \in [2, \infty)$. This means for $q \in (2, \infty)$, the same rate as attainable by the RKHS can be achieved. Now, the problem reduces to computing $\gamma_K(\mathbb{P}_m, \mathbb{Q}_n)$. Note that (9) cannot be computed in a closed form for all $q$—see the discussion in the supplementary material about approximating $\gamma_K(\mathbb{P}_m, \mathbb{Q}_n)$. However, when $q = 2$, (9) can be computed very efficiently in closed form (in terms of $K$) as a V-statistic [9], given by

$$\gamma_K^2(\mathbb{P}_m, \mathbb{Q}_n) = \sum_{j,l=1}^{m} \frac{K(X_j, X_l)}{m^2} + \sum_{j,l=1}^{n} \frac{K(Y_j, Y_l)}{n^2} - 2 \sum_{j=1}^{m} \sum_{l=1}^{n} \frac{K(X_j, Y_l)}{mn}. \qquad (10)$$

More generally, it can be shown that if $q = 2s$, $s \in \mathbb{N}$, then (9) reduces to

$$\gamma_K^q(\mathbb{P}_m, \mathbb{Q}_n) = \int_{\mathcal{X}} \overset{q}{\cdots} \int_{\mathcal{X}} \overbrace{\int_{\mathcal{X}} \prod_{j=1}^{s} b(x_{2j-1}, t)\overline{b(x_{2j}, t)}\, d\mu(t)}^{\mathcal{A}(x_1, \dots, x_q)} \prod_{j=1}^{q} d(\mathbb{P}_m - \mathbb{Q}_n)(x_j) \qquad (11)$$

for which closed form computation is possible for appropriate choices of $b$ and $\mu$. Refer to the supplementary material for the derivation of (11). For $b$ and $\mu$ as in Example 1, we have $\mathcal{A}(x_1, \dots, x_q) = (\mu(\mathbb{R}^d))^{\frac{2-p}{p}} K\left(\sum_{j=1}^{s} x_{2j-1}, \sum_{j=1}^{s} x_{2j}\right)$, while for $b$ and $\mu$ as in Example 2, we have $\mathcal{A}(x_1, \dots, x_q) = \mathcal{M}_\mu(\sum_{j=1}^{q} x_j)$. By appropriately choosing $\theta$ and $\mu$ in Example 3, we can obtain a closed form expression for $\mathcal{A}(x_1, \dots, x_q)$, which is proved in the supplementary material. Note that choosing $s = 1$ in (11) results in (10). (11) shows that $\gamma_K^q(\mathbb{P}_m, \mathbb{Q}_n)$ can be computed in a closed form in terms of $\mathcal{A}$ at a complexity of $O(m^q)$, assuming $m = n$, which means the least complexity is obtained for $q = 2$. The above discussion shows that for appropriate choices of $q$, i.e., $q \in (2, \infty)$, the RKBS embeddings in Examples 1–3 are useful in practice as $\gamma_K(\mathbb{P}_m, \mathbb{Q}_n)$ is consistent and has a closed form expression. However, the drawback of the RKBS framework is that the computation of $\gamma_K(\mathbb{P}_m, \mathbb{Q}_n)$ is more involved than its RKHS counterpart.

## 6 Conclusion & Discussion

With a motivation to study the advantages/disadvantages of generalizing Hilbert space learning algorithms to Banach spaces, in this paper, we generalized the notion of RKHS embedding of probability measures to Banach spaces, in particular RKBS that are uniformly Fréchet differentiable and uniformly convex—note that this is equivalent to generalizing a RKHS based Parzen window classifier to RKBS. While we showed that most of results in RKHS like injectivity of the embedding, consistency of the Parzen window classifier, etc., nicely generalize to RKBS yielding richer distance measures on probabilities, the generalized notion is less attractive in practice compared to its RKHS counterpart because of the computational disadvantage associated with it. Since most of the existing literature on generalizing kernel methods to Banach spaces deal with more complex algorithms than a simple Parzen window classifier that is considered in this paper, we believe that most of these algorithms may have limited practical applicability, though they are theoretically appealing. This, therefore raises an important open problem of developing computationally efficient Banach space based learning algorithms.

#### Acknowledgments

The authors thank the anonymous reviewers for their constructive comments that improved the presentation of the paper. Part of the work was done while B. K. S. was a Ph. D. student at UC San Diego. B. K. S. and G. R. G. L. acknowledge support from the National Science Foundation (grants DMS-MSPA 0625409 and IIS-1054960). K. F. was supported in part by JSPS KAKENHI (B) 22300098.

## References

[1] N. Aronszajn. Theory of reproducing kernels. *Trans. Amer. Math. Soc.*, 68:337–404, 1950.

[2] B. Beauzamy. *Introduction to Banach spaces and their Geometry*. North-Holland, The Netherlands, 1985.

[3] K. Bennett and E. Bredensteiner. Duality and geometry in svm classifier. In *Proc. $17^{th}$ International Conference on Machine Learning*, pages 57–64, 2000.

[4] A. Berlinet and C. Thomas-Agnan. *Reproducing Kernel Hilbert Spaces in Probability and Statistics*. Kluwer Academic Publishers, London, UK, 2004.

[5] R. Der and D. Lee. Large-margin classification in Banach spaces. In *JMLR Workshop and Conference Proceedings*, volume 2, pages 91–98. AISTATS, 2007.

[6] K. Fukumizu, F. R. Bach, and M. I. Jordan. Dimensionality reduction for supervised learning with reproducing kernel Hilbert spaces. *Journal of Machine Learning Research*, 5:73–99, 2004.

[7] K. Fukumizu, A. Gretton, X. Sun, and B. Schölkopf. Kernel measures of conditional dependence. In J.C. Platt, D. Koller, Y. Singer, and S. Roweis, editors, *Advances in Neural Information Processing Systems 20*, pages 489–496, Cambridge, MA, 2008. MIT Press.

[8] J. R. Giles. Classes of semi-inner-product spaces. *Trans. Amer. Math. Soc.*, 129:436–446, 1967.

[9] A. Gretton, K. M. Borgwardt, M. Rasch, B. Schölkopf, and A. Smola. A kernel method for the two sample problem. In B. Schölkopf, J. Platt, and T. Hoffman, editors, *Advances in Neural Information Processing Systems 19*, pages 513–520. MIT Press, 2007.

[10] A. Gretton, K. Fukumizu, C. H. Teo, L. Song, B. Schölkopf, and A. J. Smola. A kernel statistical test of independence. In J. Platt, D. Koller, Y. Singer, and S. Roweis, editors, *Advances in Neural Information Processing Systems 20*, pages 585–592. MIT Press, 2008.

[11] M. Hein, O. Bousquet, and B. Schölkopf. Maximal margin classification for metric spaces. *J. Comput. System Sci.*, 71:333–359, 2005.

[12] G. Lumer. Semi-inner-product spaces. *Trans. Amer. Math. Soc.*, 100:29–43, 1961.

[13] C. A. Micchelli and M. Pontil. A function representation for learning in Banach spaces. In *Conference on Learning Theory*, 2004.

[14] B. Schölkopf and A. J. Smola. *Learning with Kernels*. MIT Press, Cambridge, MA, 2002.

[15] J. Shawe-Taylor and N. Cristianini. *Kernel Methods for Pattern Analysis*. Cambridge University Press, UK, 2004.

[16] A. J. Smola, A. Gretton, L. Song, and B. Schölkopf. A Hilbert space embedding for distributions. In *Proc. 18th International Conference on Algorithmic Learning Theory*, pages 13–31. Springer-Verlag, Berlin, Germany, 2007.

[17] K. Sridharan and A. Tewari. Convex games in Banach spaces. In *Conference on Learning Theory*, 2010.

[18] B. K. Sriperumbudur, K. Fukumizu, A. Gretton, G. R. G. Lanckriet, and B. Schölkopf. Kernel choice and classifiability for RKHS embeddings of probability distributions. In Y. Bengio, D. Schuurmans, J. Lafferty, C. K. I. Williams, and A. Culotta, editors, *Advances in Neural Information Processing Systems 22*, pages 1750–1758. MIT Press, 2009.

[19] B. K. Sriperumbudur, A. Gretton, K. Fukumizu, G. R. G. Lanckriet, and B. Schölkopf. Injective Hilbert space embeddings of probability measures. In R. Servedio and T. Zhang, editors, *Proc. of the $21^{st}$ Annual Conference on Learning Theory*, pages 111–122, 2008.

[20] B. K. Sriperumbudur, A. Gretton, K. Fukumizu, B. Schölkopf, and G. R. G. Lanckriet. Hilbert space embeddings and metrics on probability measures. *Journal of Machine Learning Research*, 11:1517–1561, 2010.

[21] H. Tong, D.-R. Chen, and F. Yang. Least square regression with $\ell^p$-coefficient regularization. *Neural Computation*, 22:3221–3235, 2010.

[22] U. von Luxburg and O. Bousquet. Distance-based classification with Lipschitz functions. *Journal for Machine Learning Research*, 5:669–695, 2004.

[23] H. Wendland. *Scattered Data Approximation*. Cambridge University Press, Cambridge, UK, 2005.

[24] H. Zhang, Y. Xu, and J. Zhang. Reproducing kernel Banach spaces for machine learning. *Journal of Machine Learning Research*, 10:2741–2775, 2009.

